# A Randomized Algorithm for Large Scale Support Vector Learning

**Krishnan S.**

Department of Computer Science and Automation, Indian Institute of Science, Bangalore-12
krishi@csa.iisc.ernet.in

**Chiranjib Bhattacharyya**

Department of Computer Science and Automation, Indian Institute of Science, Bangalore-12
chiru@csa.iisc.ernet.in

**Ramesh Hariharan**

Strand Genomics, Bangalore-80
ramesh@strandls.com

## Abstract

This paper investigates the application of randomized algorithms for large scale SVM learning. The key contribution of the paper is to show that, by using ideas random projections, the minimal number of support vectors required to solve almost separable classification problems, such that the solution obtained is near optimal with a very high probability, is given by $O(\log n)$; if on removal of properly chosen $O(\log n)$ points the data becomes linearly separable then it is called almost separable. The second contribution is a sampling based algorithm, motivated from randomized algorithms, which solves a SVM problem by considering subsets of the dataset which are greater in size than the number of support vectors for the problem. These two ideas are combined to obtain an algorithm for SVM classification problems which performs the learning by considering only $O(\log n)$ points at a time. Experiments done on synthetic and real life datasets show that the algorithm does scale up state of the art SVM solvers in terms of memory required and execution time without loss in accuracy. It is to be noted that the algorithm presented here nicely complements existing large scale SVM learning approaches as it can be used to scale up any SVM solver.

## 1 Introduction

Consider a training dataset $D = \{(x_i, y_i)\}, i = 1 \ldots n$ and $y_i = \{+1, -1\}$, where $x_i \in R^d$ are data points and $y_i$ specify the class labels. the problem of learning the classifier, $y = \text{sign}(w^T x + b)$, can be narrowed down to computing $\{w, b\}$ such that it has good generalization ability. The SVM formulation for classification, which will be called $C - SVM$, for determining $\{w, b\}$ is given by [1]

**C-SVM-1:**

$$Minimize_{(w,b,\xi)} \frac{1}{2}||w||^2 + C\sum_{i=1}^{n} \xi_i$$

$$Subject\ to: \ y_i(w \cdot x_i + b) \geq 1 - \xi_i, \ , \ \xi_i \geq 0, \ i = 1 \ldots n$$

At optimality $w$ is given by $w = \sum_{i:\alpha_i>0} \alpha_i y_i x_i, \ \ 0 \leq \alpha_i \leq C$

Consider the set $S = \{x_i | \alpha_i > 0\}$; the elements of this set are called the *Support vectors*. Note that $S$ completely determines the solution of $C - SVM$. The set $S$ may not be unique, though $w$ is. Define a parameter $\Delta$ to be the minimum cardinality over all $S$. See that $\Delta$ does not change with number of examples, $n$, and is often much less than $n$.

More generally, the $C - SVM$ problem can be seen as an instance of *Abstract optimization problem*(AOP) [2, 3, 4]. An AOP is defined as follows:
*An AOP is a triple $(H, <, \Phi)$ where $H$ is a finite set, $<$ a total ordering on $2^H$, and $\Phi$ an oracle that, for a given $F \subseteq G \subseteq H$, either reports $F = min_< F' | F' \subseteq G$ or returns a set $F' \subseteq G$ with $F' < F$.*
Many SVM learning problems are AOP problems; algorithms developed for AOP problems can be used for solving SVM problems. Every AOP has a *combinatorial dimension* associated with it; the combinatorial dimension captures the notion of number of free variables for that AOP. An AOP can be solved by a randomized algorithm by selecting subsets of size greater than the *combinatorial dimension* of the problem [2].

For SVM, $\Delta$ is the combinatorial dimension of the problem; by iterating over subsets of size greater than $\Delta$, the subsets chosen using random sampling, the problem can be solved efficiently [3, 4]; this algorithm was called RandSVM by the authors. Apriori the value of $\Delta$ is not known, but for linearly separable classification problems the following holds: $2 \leq \Delta \leq d + 1$. This follows from the fact that the dual problem is the minimum distance between 2 non-overlapping convex hulls[5]. When the problem is not linearly separable, the authors use the reduced convex hull formulation [5] to come up with an estimate of the combinatorial dimension; this estimate is not very clear and much higher than $d^1$. The algorithm RandSVM[2] iterates over subsets of size proportional to $\Delta^2$.

RandSVM is not practical because of the following reasons: the sample size is too large in case of high dimensional datasets, the dimension of feature space is usually unknown when using kernels, and the reduced convex hull method used to calculate the combinatorial dimension, when the data is not separable in the feature space, isn't really useful as the number obtained is very large.

This work overcomes the above problems using ideas from random projections[6, 7] and randomized algorithms[8, 9, 2, 10],. As mentioned by the authors of RandSVM, the biggest bottleneck in their algorithm is the value of $\Delta$ as it is too large. The main contribution is, using ideas from random projections, the conjecture that if RandSVM is solved using $\Delta$ equal to $O(\log n)$, then the solution obtained is close to optimal with high probability(**Theorem 3**), in particular for *almost separable* datasets. Almost separable datasets are those which become linearly separable when a small number of properly chosen data points are deleted from them. The second contribution is an algorithm which, using ideas from randomized algorithms for Linear Programming(LP), solves the SVM problem by using samples of size linear in $\Delta$. This work also shows that the theory can be applied to non-linear kernels.

## 2 A NEW RANDOMIZED ALGORITHM FOR CLASSIFICATION

This section uses results from random projections, and randomized algorithms for linear programming, to develop a new algorithm for learning large scale SVM problems. In Section 2.1, we discuss the case of linearly separable data and estimate the number of support vectors required such that the margin is preserved with high probability, and show that this number is much smaller than the data dimension $d$, using ideas from random projections. In Section 2.2, we look how the analysis applies to *almost separable* data and present the main result of the paper(Theorem 2.2). The section ends with a discussion on the application of the theory to non-linear kernels. In Section 2.3, we present shows the randomized algorithm from SVM learning.

### 2.1 Linearly separable data

We start with determining the dimension $k$ of the target space such that on performing a random projection to the space, the Euclidean distances and dot products are preserved. The appendix contains a few results from random projections which will be used in this section.

For a linearly separable dataset $D = \{(x_i, y_i), i = 1, \ldots, n\}, x_i \in R^d, y_i \in \{+1, -1\}$, the C-SVM formulation is the same as **C-SVM-1** with $\xi_i = 0, i = 1 \ldots n$. By dividing all the constraints by $||w||$, the problem can be reformulated as follows:

**C-SVM-2a:**

$$Maximize_{(\hat{w}, b, l)} l; \ Subject \ to: \ y_i(\hat{w} \cdot x_i + \hat{b}) \geq l, \ i = 1 \ldots n, \ ||\hat{w}|| = 1$$

where $\hat{w} = \frac{w}{||w||}$, $\hat{b} = \frac{b}{||w||}$, and $l = \frac{1}{||w||}$. $l$ is the margin induced by the separating hyperplane, that is, it is the distance between the 2 supporting hyperplanes, $h1 : y_i(w \cdot x_i + b) = 1$ and $h2 : y_i(w \cdot x_i + b) = -1$.

The determination of $k$ proceeds as follows. First, for any given value of $k$, we show the change in the margin as a function of $k$, if the data points are projected onto the $k$ dimensional subspace and the problem solved. From this, we determine the value $k(k << d)$ which will preserve margin with a very high probability. In a $k$ dimensional subspace, there are at the most $k + 1$ support vectors. Using the idea of *orthogonal extensions*(definition appears later in this section), we prove that when the problem is solved in the original space, using an estimate of $k + 1$ on the number of support vectors, the margin is preserved with a very high probability.

Let $w'$ and $x'_i, i = 1, \ldots, n$ be the projection of $\hat{w}$ and $x_i, i = 1, \ldots, n$ respectively onto a $k$ dimensional subspace (as in **Lemma 2**, **Appendix A**). The classification problem in the projected space with the dataset being $D' = \{(x'_i, y_i), i = 1, \ldots, n\}, x'_i \in R^k, y_i \in \{+1, -1\}$ can be written as follows:

**C-SVM-2b:**

$$Maximize_{(w', \hat{b}, l')} l'; \ Subject \ to: \ y_i(w' \cdot x'_i + \hat{b}) \geq l', \ i = 1 \ldots n, \ ||w'|| \leq 1$$

where $l' = l(1 - \gamma)$, $\gamma$ is the distortion and $0 < \gamma < 1$. The following lemma predicts, for a given value of $\gamma$, the $k$ such that the margin is preserved with a high probability upon projection. be solved with the optimal margin obtained close to the optimal margin for the original problem is given by the following lemma.

**Theorem 1.** *Let $L = max||x_i||$ and $(w^*, b^*, l^*)$ be the optimal solution for **C-SVM-2a**. Let $R$ be a random $d \times k$ matrix as given in **Lemma 2(Appendix A)**. Let $\widetilde{w} = \frac{R^T w^*}{\sqrt{k}}$ and $x'_i = \frac{R^T x_i}{\sqrt{k}}, i = 1, \ldots, n$ and $k \geq \frac{8}{\gamma^2}(1 + \frac{(1+L^2)}{2l^*})^2 \log \frac{4n}{\delta}, \ 0 < \gamma < 1, \ 0 < \delta < 1$, then the following bound holds on the optimal margin $l_P$ obtained by solving the problem **C-SVM-2b**:*

$$P(l_P \geq l^*(1 - \gamma)) \geq 1 - \delta$$

*Proof.* From **Corollary 1** of **Lemma 2(Appendix A)**, we have

$$w^* \cdot x_i - \frac{\epsilon}{2}(1 + L^2) \leq \widetilde{w} \cdot x'_i \leq w^* \cdot x_i + \frac{\epsilon}{2}(1 + L^2)$$

which holds with probability at least $1 - 4e^{-\epsilon^2 \frac{k}{8}}$, for some $\epsilon > 0$. Consider some example $x_i$ with $y_i = 1$. Then the following holds with probability at least $1 - 2e^{-\epsilon^2 \frac{k}{8}}$

$$\widetilde{w} \cdot x'_i + b^* \geq w^* \cdot x_i - \frac{\epsilon}{2}(1 + L^2) + b^* \geq l^* - \frac{\epsilon}{2}(1 + L^2)$$

Dividing the above by $||\widetilde{w}||$, we have $\frac{\widetilde{w} \cdot x'_i + b^*}{||\widetilde{w}||} \geq \frac{l^* - \frac{\epsilon}{2}(1+L^2)}{||\widetilde{w}||}$. Note that from **Lemma 1(Appendix A)**, we have $(1 - \epsilon)||w^*|| \leq ||\widetilde{w}|| \leq (1 + \epsilon)||w^*||$, with probability at least $1 - 2e^{-\epsilon^2 \frac{k}{8}}$. Since $||w^*|| = 1$, we have $\sqrt{1 - \epsilon} \leq ||\widetilde{w}|| \leq \sqrt{1 + \epsilon}$. Hence we have

$$
\begin{aligned}
\frac{\widetilde{w} \cdot x'_i + b^*}{||\widetilde{w}||} &\geq \frac{l^* - \frac{\epsilon}{2}(1 + L^2)}{\sqrt{1 + \epsilon}} \\
&\geq (l^* - \frac{\epsilon}{2}(1 + L^2))(\sqrt{1 - \epsilon}) = l^*(1 - \frac{\epsilon}{2l^*}(1 + L^2)(\sqrt{1 - \epsilon})) \\
&\geq l^*(\sqrt{1 - \epsilon} - \frac{\epsilon}{2l^*}(1 + L^2)) = l^*(1 - \epsilon(1 + \frac{1 + L^2}{2l^*}))
\end{aligned}
$$

This holds with probability at least $1 - 4e^{-\epsilon^2 \frac{k}{8}}$. A similar result can be derived for a point $x_j$ for which $y_j = -1$. The above analysis guarantees that by projecting onto a $k$ dimensional space, there exists at least one hyperplane $(\frac{\widetilde{w}}{||\widetilde{w}||}, \frac{b^*}{||\widetilde{w}||})$, which guarantees a margin of $l^*(1 - \gamma)$ where

$$\gamma \leq \epsilon(1 + \frac{1 + L^2}{2l^*}) \tag{1}$$

with probability at least $1 - n4e^{-\epsilon^2 \frac{k}{8}}$. The margin obtained by solving the problem **C-SVM-2b**, $l_P$ can only be better than this. So the value of $k$ is given by:

$$n4e^{-\frac{\gamma^2}{(1 + \frac{1 + L^2}{2l^*})^2} \frac{k}{8}} \leq \delta \implies k \geq \frac{8(1 + \frac{(1 + L^2)}{2l^*})^2}{\gamma^2} \log \frac{4n}{\delta} \tag{2}$$

As seen above, by randomly projecting the points onto a $k$ dimensional subspace, the margin is preserved with a high probability. This result is similar to the results obtained in work on random projections[7]. But there are fundamental differences between the method proposed in this paper and the previous methods: No random projection is actually done here, and no black box access to the data distribution is required. We use **Theorem 1** to determine an estimate on the number of support vectors such that margin is preserved with a high probability, when the problem is solved in the original space. This is given in **Theorem 2** and is the main contribution of this section. The theorem is based on the following fact: in a $k$ dimensional space, the number of support vectors is upper bounded by $k + 1$. We show that this $k + 1$ can be used as an estimate of the number of support vectors in the original space such that the solution obtained preserves the margin with a high probability. We start with the following definition.

**Definition** *An* `orthogonal extension` *of a $k - 1$-dimensional flat( a $k - 1$ dimensional flat is a $k - 1$-dimensional affine space) $h_p = (w_p, b)$, where $w_p = (w_1, \ldots, w_k)$, in a subspace $S_k$ of dimension $k$ to a $d - 1$-dimensional hyperplane $h = (\widetilde{w}, b)$ in $d$-dimensional space, is defined as follows. Let $R \in R^{d \times d}$ be a random projection matrix as in **Lemma 2**((Appendix A)). Let $\hat{R} \in R^{d \times k}$ be a another random projection matrix which consists of only the the first $k$ columns of $R$. Let $\hat{x}_i = R^T x_i$ and $x_i' = \frac{\hat{R}^T}{\sqrt{k}} x_i$ as follows: Let $w_p = (w_1, \ldots, w_k)$ be the optimal hyperplane classifier with margin $l_P$ for the points $x_1', \ldots, x_n'$ in the $k$ dimensional subspace. Now define $\widetilde{w}$ to be all 0's in the last $d - k$ coordinates and identical to $w_p$ in the first $k$ coordinates, that is, $\widetilde{w} = (w_1, \ldots, w_k, 0, \ldots, 0)$. Orthogonal extensions have the following key property. If $(w_p, b)$ is a separator with margin $l_p$ for the projected points, then its orthogonal extension $(\widetilde{w}, b)$ is a separator with margin $l_p$ for the original points,that is,*
*if, $y_i(w_p \cdot x_i' + b) \geq l, i = 1, \ldots, n$ then $y_i(\widetilde{w} \cdot \hat{x}_i + b) \geq l, i = 1, \ldots, n$*

An important point to note, which will be required when extending orthogonal extensions to non-linear kernels, is that dot products between the points are preserved upon doing orthogonal projections, that is, $x_i'^T x_j' = \hat{x}_i^T \hat{x}_j$.

Let $L, l^*, \gamma, \delta$ and $n$ be as defined in **Theorem 1**. The following is the main result of this section.

**Theorem 2.** *Given $k \geq \frac{8}{\gamma^2}(1 + \frac{(1 + L^2)}{2l^*})^2 \log \frac{4n}{\delta}$ and $n$ training points with maximum norm $L$ in $d$ dimensional space and separable by a hyperplane with margin $l^*$, there exists a subset of $k'$ training points $x_{1'} \ldots x_{k'}$ where $k' \leq k$ and a hyperplane $h$ satisfying the following conditions:*

  1. *$h$ has margin at least $l^*(1 - \gamma)$ with probability at least $1 - \delta$*

  2. *$x_{1'} \ldots x_{k'}$ are the only training points which lie either on $h_1$ or on $h_2$*

*Proof.* Let $w^*, b^*$ denote the normal to a separating hyperplane with margin $l^*$, that is, $y_i(w^* \cdot x_i + b^*) \geq l^*$ for all $x_i$ and $||w^*|| = 1$. Consider a random projection of $x_1, \ldots, x_n$ to a $k$ dimensional space and let $w', z_1, \ldots, z_n$ be the projections of $w^*, x_1, \ldots, x_n$, respectively, scaled by $1/\sqrt{k}$. By **Theorem 1**, $y_i(w' \cdot z_i + b^*) \geq l^*(1 - \gamma)$ holds for all $z_i$ with probability at least $1 - \delta$. Let $h$ be the orthogonal extension of $w', b^*$ to the full $d$ dimensional space. Then $h$ has margin at least $l^*(1 - \gamma)$, as required. This shows the first part of the claim.

To prove the second part, consider the projected training points which lie on $w', b^*$ (that is, they lie on either of the two sandwiching hyperplanes). Barring degeneracies, there are at the most $k$ such points. Clearly, these will be the only points which lie on the orthogonal extension $h$, by definition.□

From the above analysis, it is seen that if $k << d$, then we can estimate that the number of support vectors is $k + 1$, and the algorithm RandSVM would take on average $O(k \log n)$ iterations to solve the problem [3, 4].

## 2.2 Almost separable data

In this section, we look at how the above analysis can be applied to *almost separable* datasets. We call a dataset *almost separable* if by removing a fraction $\kappa = O(\frac{\log n}{n})$ of the points, the dataset becomes linearly separable.

The C-SVM formulation when the data is not linearly separable(and *almost separable*) was given in **C-SVM-1**. This problem can be reformulated as follows:

$$Minimize_{(w,b,\xi)} \sum_{i=1}^{n} \xi_i$$

$$Subject\ to:\ y_i(w \cdot x_i + b) \geq l - \xi_i,\ \xi_i \geq 0,\ i = 1 \ldots n; ||w|| \leq \frac{1}{l}$$

This formulation is known as the *Generalized Optimal Hyperplane* formulation. Here $l$ depends on the value of $C$ in the C-formulation. At optimality, the margin $l^* = l$. The following theorem proves a result for almost separable data similar to the one proved in **Claim 1** for separable data.

**Theorem 3.** *Given* $k \geq \frac{8}{\gamma^2}(1 + \frac{(1+L^2)}{2l^*})^2 \log \frac{4n}{\delta}\ + \kappa n$, $l^*$ *being the margin at optimality,* $l$ *the lower bound on* $l^*$ *as in the Generalized Optimal Hyperplane formulation and* $\kappa = O(\frac{\log n}{n})$, *there exists a subset of* $k'$ *training points* $x_{1'} \ldots x_{k'}$, $k' \leq k$ *and a hyperplane* $h$ *satisfying the following conditions:*

1. *h has margin at least* $l(1 - \gamma)$ *with probability at least* $1 - \delta$

2. *At the most* $\frac{8(1+\frac{(1+L^2)}{2l^*})^2}{\gamma^2} \log \frac{4n}{\delta}$ *points lie on the planes* $h_1$ *or on* $h_2$

3. $x_{1'}, \ldots, x_{k'}$ *are the only points which define the hyperplane* $h$, *that is, they are the support vectors of* $h$.

*Proof.* Let the optimal solution for the generalized optimal hyperplane formulation be $(w^*, b^*, \xi^*)$. $w^* = \sum_{i:\alpha_i > 0} \alpha_i y_i x_i$, and $l^* = \frac{1}{||w^*||}$ as mentioned before. The set of support vectors can be split into to 2 disjoint sets,$SV_1 = \{x_i : \alpha_i > 0 \text{ and } \xi_i^* = 0\}$(unbounded SVs), and $SV_2 = \{x_i : \alpha_i > 0 \text{ and } \xi_i^* > 0\}$(bounded SVs).

Now, consider removing the points in $SV_2$ from the dataset. Then the dataset becomes linearly separable with margin $l^*$. Using an analysis similar to **Theorem 1**, and the fact that $l^* \geq l$, we have the proof for the first 2 conditions.

When all the points in $SV_2$ are added back, at most all these points are added to the set of support vectors and the margin does not change. The margin not changing is guaranteed by the fact that for proving the conditions 1 and 2, we have assumed the worst possible margin, and any value lower than this would violate the constraints of the problem. This proves condition 3. □

Hence the number of support vectors, such that the margin is preserved with high probability, can be upper bounded by

$$k + 1 = \frac{8}{\gamma^2}(1 + \frac{(1 + L^2)}{2l^*})^2 \log \frac{4n}{\delta}\ + \kappa n + 1 = \frac{8}{\gamma^2}(1 + \frac{(1 + L^2)}{2l^*})^2 \log \frac{4n}{\delta}\ + O(\log n) \quad (3)$$

**Using a non-linear kernel**   Consider a mapping function $\Phi : R^d \to R^{d'}$, $d' > d$, which projects a point $x_i \in R^d$ to a point $z_i \in R^{d'}$, where $R^{d'}$ is a Euclidean space. Let the points be projected onto a random $k$ dimensional subspace as before. Then, as in the case of linear kernels, the lemmata in the appendix are applicable to these random projections[11]. The orthogonal extensions can be

considered as a projection from the $k$ dimensional space to the $\Phi$-space, such that the kernel function values are preserved. Then it can be shown that **Theorem 3** applies when using non-linear kernels also.

## 2.3 A Randomized Algorithm

The reduction in the sample size from $6d^2$ to $6k^2$ is not enough to make RandSVM useful in practice as $6k^2$ is still a large number. This section presents another randomized algorithm which only requires that the sample size be greater than the number of support vectors. Hence a sample size linear in $k$ can be used in the algorithm. This algorithm was first proposed to solve large scale LP problems[10]; it has been adapted for solving large scale SVM problems.

---

**Algorithm 1** RandSVM-1(D,k,r)

---
**Require:** $D$ - The dataset.
**Require:** $k$ - The estimate of the number of support vectors.
**Require:** $r$ - Sample size $= ck, c > 0$.
1: $S = \text{randomsubset}(D, r)$; // *Pick a random subset, S, of size r from the dataset D*
2: $SV = \text{svmlearn}(\Phi, S)$; // *SV - set of support vectors obtained by solving the problem S*
3: $V = \{x \in D - S | violates(x, SV)\}$ //*violator - nonsampled point not satisfying KKT conditions*
4: **while** $|V| > 0$ and $|SV| < k$ **do**
5:    $R = \text{randomsubset}(V, r - |SV|)$; //*Pick a random subset from the set of violators*
6:    $SV = \text{svmlearn}(SV, R)$; //*SV - set of support vectors obtained by solving the problem $SV \cup R$*
7:    $V = \{x \in D - (SV \cup R) | violates(x, SV)\}$; //*Determine violators from nonsampled set*
8: **end while**
9: return $SV$

---

**Proof of Convergence:** Let $SV$ be the current set of support vectors. Condition $|SV| < k$ comes from **Theorem 3**. Hence if the condition is violated, then the algorithm terminates solution which is near optimal with a very high probability.
Now consider the case where $|SV| < k$ and $|V| > 0$. Let $x_i$ be a violator($x_i$ is a non-sampled point such that $y_i(w^T x_i + b) < 1$). Solving the problem with the set of constraints as $SV \cup x_i$ will only result, since SVM is an instance of AOP, in the increase(decrease) of the objective function of the primal(dual). As there are only finite number of basis for an AOP, the algorithm is bound to terminate; also if termination happens with the number of violators equal to zero, then the solution obtained is optimal.

**Determination of $k$**  The value of $k$ depends on the $l$ which is not available in case of $C$-SVM and $nu$-SVM. This can be handled only be solving for $k$ as a function of $\epsilon$ where $\epsilon$ is the maximum allowed distortion in the $L_2$ norms of the vectors upon projection. If all the data points are normalized to length 1, that is, $L = 1$, then Equation 1 becomes $\epsilon \geq \gamma/(1 + \frac{1+L^2}{2l^*})$. Combining this with the result from **Theorem 2**, the value of $k$ can be determined in terms of $\epsilon$ as follows:

$$k \geq \frac{8}{\gamma^2}(1 + \frac{(1+L^2)}{2l^*})^2 \log \frac{4n}{\delta} + O(\log n) \geq \frac{16}{\gamma^2}(1 + \frac{(1+L^2)}{2l^*})^2 \log \frac{4n}{\delta}) \geq \frac{16}{\epsilon^2} \log \frac{4n}{\delta} \quad (4)$$

# 3 Experiments

This section discusses the performance of RandSVM in practice. The experiments were performed on 3 synthetic and 1 real world dataset. RandSVM was used with LibSVM as the solver when using a non-linear kernel; with SVMLight for a linear kernel. This choice was made because it was observed that SVMLight is much faster than LibSVM when using a linear kernel, and vice-versa when using non-linear kernels. RandSVM has been compared with state of the art SVM solvers: LibSVM for non-linear kernels, and SVMPerf and SVMLin for linear kernels.
**Synthetic datasets**
The twonorm dataset is a 2 class problem where each class is drawn from a multivariate normal distribution with unit variance. Each vector is a 20 dimensional vector. One class has mean $(a, a, \ldots, a)$, and the other class has mean $(-a, -a, \ldots, -a)$, where $a = 2/\sqrt{(20)}$.
The ringnorm dataset is a 2 class problem with each vector consisting of 20 dimensions. Each class

| Category | Kernel | RandSVM | LibSVM | SVMPerf | SVMLin |
|---|---|---|---|---|---|
| twonorm$_1$ | Gaussian | 300 (94.98%) | 8542 (96.48%) | X | X |
| twonorm$_2$ | Gaussian | 437 (94.71%) | - | X | X |
| ringnorm$_1$ | Gaussian | 2637 (70.66%) | 256 (70.31%) | X | X |
| ringnorm$_2$ | Gaussian | 4982 (65.74%) | 85124 (65.34%) | X | X |
| checkerboard$_1$ | Gaussian | 406 (93.70%) | 1568.93 (96.90%) | X | X |
| checkerboard$_2$ | Gaussian | 814 (94.10%) | - | X | X |
| CCAT* | Linear | 345 (94.37%) | X | 148 (94.38%) | 429(95.1913%) |
| C11* | Linear | 449 (96.57%) | X | 120 (97.53%) | 295 (97.71%) |

Table 1: The table gives the execution time(in seconds) and the classification accuracy(in brackets). The subscripts 1 and 2 indicate that the corresponding training set sizes are $10^5$ and $10^6$ respectively. A '-' indicates that the solver did not finish execution even after a running for a day. A 'X' indicates that the experiment is not applicable for the corresponding solver. The '*' indicates that the solver used with RandSVM was SVMLight; otherwise it was LibSVM.

is drawn from a multivariate normal distribution. One class has mean 1, and covariance 4 times the identity. The other class has mean $(a, a, \ldots, a)$, and unit covariance where $a = 2/\sqrt{(20)}$.

The checkerboard dataset consists of vectors in a 2 dimensional space. The points are generated in a $4 \times 4$ grid. Both the classes are generated from a multivariate uniform distribution; each point is $(x1 = U(0, 4), x2 = U(0, 4))$. The points are labelled as follows - if($\lceil x1 \rceil \% 2 \neq \lceil x2 \rceil \% 2$), then the point is labelled negative, else the point is labelled positive.

For each of the synthetic datasets, a training set of 10,00,000 points and a test set of 10,000 points was generated. A smaller subset of 1,00,000 points was chosen from training set for parameter tuning. From now on, the smaller training set will have a subscript of 1 and the larger training set will have a subscript of 2, for example, ringnorm$_1$ and ringnorm$_2$.

**Real world dataset**

The RCV1 dataset consists of 804,414 documents, with each document consisting of 47,236 features. Experiments were performed using 2 categories of the dataset - CCAT and C11. The dataset was split into a training set of 7,00,000 documents and a test set of 104,414 documents.

Table 1 shows the kernels which were used for each of the datasets. The parameters used for the gaussian kernels, $\sigma$ and $C$, were obtained using grid search based tuning. The parameter for the linear kernel, $C$, for CCAT and C11 were obtained from previous work done[12].

**Selection of $k$ for RandSVM:** The values of $\epsilon$ and $\delta$ were fixed to 0.2 and 0.9 respectively, for all the datasets. For linearly separable datasets, $k$ was set to $(16 \log(4n/\delta))/\epsilon^2$. For the others, $k$ was set to $(32 \log(4n/\delta))/\epsilon^2$.

**Discussion of results:** Table 1, which has the timing and classification accuracy comparisons, shows that RandSVM can scale up SVM solvers for very large datasets. Using just a small wrapper around the solvers, RandSVM has scaled up SVMLight so that its performance is comparable to that of state of the art solvers such as SVMPerf and SVMLin. Similarly LibSVM has been made capable of quickly solving problems which it could not do before, even after executing for a day. Furthermore, it is clear, from the experiments on the synthetic datasets, that the execution times taken for training with $10^5$ examples and $10^6$ examples are not too far apart; this is a clear indication that the execution time does not increase rapidly with the increase in the dataset size.

All the runs of RandSVM terminated with the condition $|SV| < k$ being violated. Since the classification accuracies obtained by using RandSVM and the baseline solvers are very close, it is clear that **Theorem 2** holds in practice.

## 4   Further Research

It is clear from the experimental evaluations that randomized algorithms can be used to scale up SVM solvers to large scale classification problems. If an estimate of the number of support vectors is obtained then algorithm RandSVM-1 can be used for other SVM learning problems also, as they are usually instances of an AOP. The future work would be to apply the work done here to such problems.

## A   Some Results from Random Projections

Here we review a few lemmas from random projections [7]. The following lemma discusses how the $L_2$ norm of a vector is preserved when it is projected on a random subspace.

**Lemma 1.** *Let $R = (r_{ij})$ be a random $d \times k$ matrix, such that each entry $(r_{ij})$ is chosen independently according to $N(0,1)$. For any fixed vector $u \in R^d$, and any $\epsilon > 0$, let $u' = \frac{R^T u}{\sqrt{k}}$. Then $E[||u'||^2] = ||u||^2$ and the following bound holds:*

$$P((1-\epsilon)||u||^2 \leq ||u'||^2 \leq (1+\epsilon)||u||^2) \geq 1 - 2e^{-(\epsilon^2 - \epsilon^3)\frac{k}{4}}$$

The following theorem and its corollary show the change in the Euclidean distance between 2 points and the dot products when they are projected onto a lower dimensional space [7].

**Lemma 2.** *Let $u, v \in R^d$. Let $u' = \frac{R^T u}{\sqrt{k}}$ and $v' = \frac{R^T u}{\sqrt{k}}$ be the projections of $u$ and $v$ to $R^k$ via a random matrix $R$ whose entries are chosen independently from $N(0,1)$ or $U(-1,1)$. Then for any $\epsilon > 0$, the following bounds hold*

$$P((1-\epsilon)||u-v||^2 \leq ||u'-v'||^2) \geq 1 - e^{-(\epsilon^2-\epsilon^3)\frac{k}{4}}, \text{ and}$$
$$P(||u'-v'||^2 \leq (1+\epsilon)||u-v||^2) \geq 1 - e^{-(\epsilon^2-\epsilon^3)\frac{k}{4}}$$

A corollary of the above theorem shows how well the dot products are preserved upon projection(This is a slight modification of the corollary given in [7]).

**Corollary 1.** *Let $u, v$ be vectors in $R^d$ s.t. $||u|| \leq L_1, ||v|| \leq L_2$. Let $R$ be a random matrix whose entries are chosen independently from either $N(0,1)$ or $U(-1,1)$. Define $u' = \frac{R^T u}{\sqrt{k}}$ and $v' = \frac{R^T v}{\sqrt{k}}$. Then for any $\epsilon > 0$, the following holds with probability at least $1 - 4e^{-\epsilon^2\frac{k}{8}}$*

$$u \cdot v - \frac{\epsilon}{2}(L_1^2 + L_2^2) \leq u' \cdot v' \leq u \cdot v + \frac{\epsilon}{2}(L_1^2 + L_2^2)$$

## Footnotes

[1] Details of this calculation are present in the supplementary material

[2] Presented in supplementary material

## References

[1] V. Vapnik. *The Nature of Statistical Learning Theory*. Springer, New York, 1995.

[2] Bernd Gartner. A subexponential algorithm for abstract optimization problems. In *Proceedings 33rd Symposium on Foundations of Computer Science, IEEE CS Press*, 1992.

[3] Jose L. Balcazar, Yang Dai, and Osamu Watanabe. A random sampling technique for training support vector machines. In *ALT*. Springer, 2001.

[4] Jose L. Balcazar, Yang Dai, and Osamu Watanabe. Provably fast training algorithms for support vector machines. In *ICDM*, pages 43–50, 2001.

[5] K. P. Bennett and E. J. Bredensteiner. Duality and geometry in SVM classifiers. In P. Langley, editor, *ICML*, pages 57–64, San Francisco, California, 2000.

[6] W. Johnson and J. Lindenstauss. Extensions of lipschitz maps into a hilbert space. *Contemporary Mathematics*, 1984.

[7] R. I. Arriaga and S. Vempala. An algorithmic theory of learning: Random concepts and random projections. In *Proceedings of the 40th Foundations of Computer Science*, 1999.

[8] Kenneth L. Clarkson. Las vegas algorithms for linear and integer programming when the dimension is small. *Journal of the ACM*, 42(2):488–499, 1995.

[9] B. Gartner and E. Welzl. A simple sampling lemma: analysis and application in geometric optimization. In *Proceedings of the 16th annual ACM symposium on Computational Geometry*, 2000.

[10] M. Pellegrini. Randomizing combinatorial algorithms for linear programming when the dimension is moderately high. In *SODA '01*, pages 101–108, Philadelphia, PA, USA, 2001.

[11] Maria-Florina Balcan, Avrim Blum, and Santosh Vempala. On kernels, margins and low-dimensional mappings. In *Proc. of the 15th Conf. Algorithmic Learning Theory*, 2004.

[12] T. Joachims. Training linear svms in linear time. In *Proceedings of the ACM Conference on Knowledge Discovery and Data Mining (KDD)*, 2006.

